# Perturbative M-Sequences for Auditory Systems Identification

**Mark Kvale and Christoph E. Schreiner***
Sloan Center for Theoretical Neurobiology, Box 0444
University of California, San Francisco
513 Parnassus Ave, San Francisco, CA 94143

## Abstract

In this paper we present a new method for studying auditory systems based on m-sequences. The method allows us to perturbatively study the linear response of the system in the presence of various other stimuli, such as speech or sinusoidal modulations. This allows one to construct linear kernels (receptive fields) at the same time that other stimuli are being presented. Using the method we calculate the modulation transfer function of single units in the inferior colliculus of the cat at different operating points and discuss nonlinearities in the response.

## 1 Introduction

A popular approach to systems identification, i.e., identifying an accurate analytical model for the system behavior, is to use Volterra or Wiener expansions to model behavior via functional Taylor or orthogonal polynomial series, respectively [Marmarelis and Marmarelis1978]. Both approaches model the response $r(t)$ as a linear combination of small powers of the stimulus $s(t)$. Although effective for mild nonlinearities, deriving the linear combinations becomes numerically unstable for highly nonlinear systems. A more serious problem is that many biological systems are adaptive, i.e., the system behavior is dependent on the stimulus ensemble. For instance, [Rieke *et al.*1995] found that in the auditory nerve of the bullfrog linearity and information rates depended sensitively on whether a white noise or naturalistic ensemble is used.

One approach to handling these difficulties is to forgo the full expansion, and simply compute the linear response to small (perturbative) stimuli in the presence of various different ensembles, or operating points. By collecting linear responses

from different operating points, one may fit nonlinear responses as one fits a nonlinear function with a piecewise linear approximation. For adaptive systems the same procedure would be applied, with different operating points corresponding to different points along the time axis. Perturbative stimuli have wide application in condensed-matter physics, where they are used to characterize linear responses such as resistance, elasticity and viscosity, and in engineering, perturbative analyses are used in circuit analysis (small signal models) and structural diagnostics (vibration analysis). In neurophysiology, however, perturbative stimuli are unknown.

An effective stimulus for calculating the perturbative linear response of a system is the m-sequence. M-sequences have a long history of use in engineering and the physical sciences, with applications ranging from systems identification to cryptography and cellular communication. In physiology, m-sequences have been used primarily to compute system kernels [Marmarelis and Marmarelis1978], especially in the visual system [Pinter and Nabet1987]. In this work, we use perturbative m-sequences to study the linear response of single units in the inferior colliculus of a cat to amplitude-modulated (AM) stimuli. We add a small m-sequence signal to an AM carrier, which allows us to study the linear behavior of the system near a particular operating point in a non-destructive manner, i.e., without changing the operating point. Perturbative m-sequences allow one to calculate linear responses near the particular stimuli under study with only a little extra effort, and allow us to characterize the system over a wide range of stimuli, such as sinusoidal AM and naturalistic stimuli.

The auditory system we selected to study was the response of single units in the central nucleus of the inferior colliculus (IC) of an anaesthetised cat. Single unit responses were recorded extracellularly. Action potentials were amplified and stored on DAT tape, and were discriminated offline using a commercial computer-based spike sorter (Brainwave). 20 units were recorded, of which 10 yielded sufficiently stable responses to be analyzed.

## 2   M-Sequences and Linear Systems

A binary m-sequence is a two-level pseudo-random sequence of +1's and −1's. The sequence length is $L = 2^n - 1$, where $n$ is the order of the sequence. Typically, a binary m-sequence can be generated by a shift register with $n$ bits and feedback connections derived from an irreducible polynomial over the multiplicative group $Z_2$ [Golomb1982]. For linear systems identification, m-sequences have two important properties. The first is that m-sequences have nearly zero mean: $\sum_{t=0}^{L-1} m[t] = -1$. The second is that the autocorrelation function takes on the impulse-like form

$$S_{mm}(\tau) = \sum_{t=0}^{L-1} m[t]m[t+\tau] = \left\{ \begin{array}{ll} L & \text{if } \tau = 0 \\ -1 & \text{otherwise} \end{array} \right. \tag{1}$$

Impulse stimuli also have a $\delta$-function autocorrelation function. In the context of perturbative stimuli, the advantage of an m-sequence stimulus over an impulse stimulus is that for a given signal to noise ratio, an m-sequence perturbation stays much closer to the original signal (in the least squares sense) than an impulse perturbation. Thus the perturbed signal does not stray as far from the operating point and measurement of linear response about that operating point is more accurate.

We model the IC response with a system $F$ through which a scalar stimulus $s(t)$ is passed to give a response $r(t)$:

$$r(t) = F[s(t)]. \tag{2}$$

For the purposes of this section, the functional $F$ is taken to be a linear functional plus a DC component. In real experiments, the input and output signal are sampled into discrete sequences with $t$ becoming an integer indexing the sequence. Then the system can be written as the discrete convolution

$$r[t] = h_0 + \sum_{t_1=0}^{L-1} h[t_1]s[t - t_1] \tag{3}$$

with kernels $h_0$ and $h[t_1]$ to be determined. We assume that the system has a finite memory of $M$ time steps (with perhaps a delay) so that at most $M$ of the $h[t]$ coefficients are nonzero. To determine the kernels perturbatively, we add a small amount of m-sequence to a base stimulus $s_0$:

$$s[t] = s_0[t] + \alpha m[t]. \tag{4}$$

Cross-correlating the response with the original m-sequence yields

$$R_{rm}(\tau) = \sum_{t=0}^{L-1} m[t]r[t + \tau] = \sum_{t=0}^{L-1} m[t]h_0 + \sum_{t=0}^{L-1}\sum_{t_1=0}^{L-1} h[t_1]m[t]s_0[t + \tau - t_1]$$

$$+ \sum_{t=0}^{L-1}\sum_{t_1=0}^{L-1} \alpha h[t_1]m[t]m[t + \tau - t_1]. \tag{5}$$

Using the sum formula for am -sequence above, the first sum in Eq. (5) can be simplified to $-h_0$. Using the autocorrelation Eq. (1), the third sum in Eq. (5) simplifies, and we find

$$R_{rm}(\tau) = \alpha(L + 1)h[\tau] - h_0 - \alpha \sum_{t_1=0}^{L-1} h[t_1] + \sum_{t=0}^{L-1}\sum_{t_1=0}^{L-1} h[t_1]m[t]s_0[t + \tau - t_1] \tag{6}$$

Although the values for the kernels $h(t)$ are set implicitly by this equation, the terms on the right hand side of Eq. (6) are widely different in size for large $L$ and the equation can be simplified. As is customary in auditory systems, we assume the DC response $h_0$ is small. To estimate the size of the other terms, we compute statistical estimates of their sizes and look at their scaling with the parameters. The term $\alpha \sum_{t_1=0}^{L-1} h[t_1]$ is a sum of $M$ kernel elements; they may be correlated or uncorrelated, so a conservative estimate of their size is on the order of $O(\alpha M)$.

The last term in (6) is more subtle. We rewrite it as

$$\sum_{t_1=0}^{L-1}\sum_{t=0}^{L-1} h[t_1]m[t]s_0[t + \tau - t_1] = \sum_{t_1=0}^{L-1} h[t_1]p[\tau, t_1]$$

$$p[\tau, t_1] = \sum_{t=0}^{L-1} m[t]s_0[t + \tau - t_1] \tag{7}$$

The time series of the ambient stimulus $s_0[t]$ and m-sequence $m[t]$ are assumed to be uncorrelated. By the central limit theorem, the sum $p[\tau, t_1]$ will then have an average of zero with a standard deviation of $O(L^{1/2})$. If in turn, the terms $p[\tau, t_1]$ are uncorrelated with the kernels $h[t_1]$, we have that

$$\sum_{t_1=0}^{L-1}\sum_{t=0}^{L-1} h[t_1]m[t]s_0[t + \tau - t_1] \sim O(M^{1/2}L^{1/2}) \tag{8}$$

If $N$ cycles of the m-sequence are performed, in which $s_0[t]$ is different for each cycle, all the terms in Eq. (6) scale with $N$ as $O(N)$, except for the double sum. By the same central limits arguments above, the double sum scales as $O(N^{1/2})$.

Putting all these results together into Eq. (6) and solving for the kernels yields

$$
\begin{aligned}
h(\tau) &= \frac{1}{\alpha(L+1)} R_{rm}(\tau) - O\left(\frac{M}{L}\right) + O\left(\frac{M^{1/2}}{\alpha N^{1/2} L^{1/2}}\right). \\
&\approx \frac{1}{\alpha(L+1)} R_{rm}(\tau) - C_1 \frac{M}{L} + C_2 \frac{M^{1/2}}{\alpha N^{1/2} L^{1/2}},
\end{aligned}
\tag{9}
$$

with the constants $C_1, C_2 \sim O(h[\tau])$ depending neural firing rate, statistics, etc., determined from experiment. If we take the kernel element $h(\tau)$ to be the first term in Eq. 9, then the last two terms in Eq. (9) contribute errors in determining the kernel and can be thought of as noise. Both error terms vanish as $L \to \infty$ and the procedure is asymptotically exact for arbitrary uncorrelated stimuli $s_0[t]$. In order for the cross-correlation $R_{sm}(\tau)$ to yield a good estimate, the inequalities

$$
C_1 M \ll L \quad \text{and} \quad \alpha \gg C_2 M^{1/2}(NL)^{-1/2}
\tag{10}
$$

must hold. In practice, the kernel memory is much smaller than the sequence length, and the second inequality is the stricter bound. The second inequality represents a tradeoff among sequence length, number of trials and the size of the perturbation for a given level of systematic noise in the kernel estimate. For instance, if $L = 2^{15} - 1$, $N = 10$, $M = 30$, and noise floor at 10%, the perturbation should be larger than $\alpha = 0.095$. If no signal $s_0[t]$ is present, then the $O(M^{1/2}\alpha^{-1}(NL)^{-1/2})$ term drops out and the usual m-sequence cross-correlation result is recovered.

## 3  M-Sequences for Modulation Response

Previous work, e.g., [Møller and Rees1986, Langner and Schreiner1988] has shown that many of the cells in the inferior colliculus are tuned not only to a characteristic frequency, but are also tuned to a best frequency of modulation of the carrier. A highly simplified model of the IC unit response to sound stimuli is the $L1 - N - L2$ cascade filter, with $L1$ a linear tank circuit with a transfer function matching that of the frequency tuning curve, $N$ a nonlinear rectifying unit, and $L2$ a linear circuit with a transfer function matching that of the modulation transfer function. Detecting this modulation is an inherently nonlinear operation and $N$ is not well approximated by a linear kernel. Thus IC modulation responses will not be well characterized by ordinary m-sequence stimuli using the methods described in Section 2.

A better approach is to bypass the $L1 - N$ demodulation step entirely and concentrate on measuring $L2$. This can be accomplished by creating a *modulation m-sequence*:

$$
s[t] = a\left(s_0[t] + b\, m[t]\right) \sin[\omega_c t],
\tag{11}
$$

where $|s_0[t]| \leq 1$ is the ambient signal, i.e., the operating point, $m[t] \in [-1, 1]$ is an m-sequence added with amplitude $b$, and $\omega_c$ is the carrier frequency. Demodulation gives the effective input stimulus

$$
s_m[t] = a\left(s_0[t] + b\, m[t]\right).
\tag{12}
$$

Note that there is little physiological evidence for a purely linear rectifier $N$. In fact, both the work of [Møller and Rees1986, Rees and Møller1987] and ours below show that there is a nonlinear modulation response. Taking a modulation transfer

function seriously, however, implies that one assumes that modulation response is linear, which implies that the static nonlinearity used is something like a half-wave rectifier. Linearity is used here as a convenient assumption for organizing the stimulus and asking whether nonlinearities exist.

For full m-sequence modulation ($s_0[t] = 1$ and $b = 1$) the stimulus $s_m$ and the neural response can be used to compute, via the Lee-Schetzen cross-correlation, the modulation transfer function for the $L2$ system. Alternatively, for $b \ll 1$, the m-sequence is a perturbation on the underlying modulation envelope $s_0[t]$. The derivation above shows that the linear modulation kernel can also be calculated using a Lee-Schetzen cross-correlation. M-sequences at full modulation depth were first used by [Møller and Rees1986, Rees and Møller1987] to calculate white-noise kernels. Here, we are using m-sequence in a different way—we are calculating the small-signal properties around the stimulus $s_0[t]$.

The m-sequences used in this experiment were of length $2^{15} - 1 = 32,767$. For each unit, 10 cycles of the m-sequence were presented back-to-back. After determining the characteristic frequency of a unit, stimuli were presented which never differed from the characteristic frequency by more than 500 Hz. Figure 1 depicts the sinusoidal and m-sequence components and their combined result. The stimuli were presented in random order so as to mitigate adaptation effects.

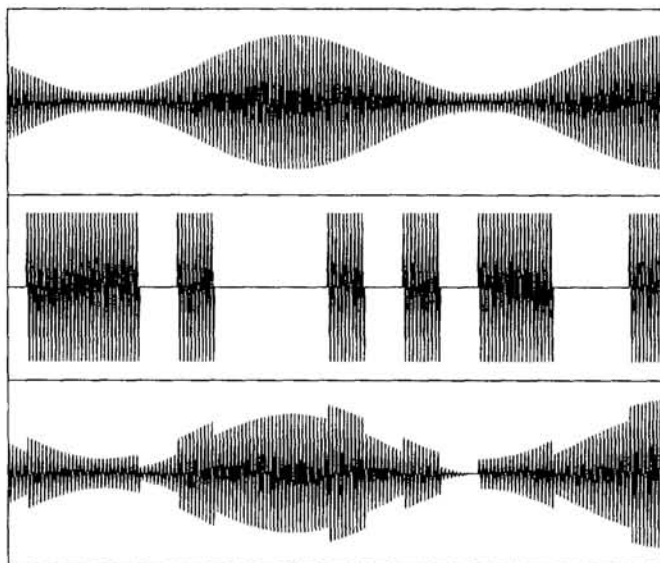

Figure 1: A depiction of stimuli used in the experiment. The top graph shows a pure sine wave modulation at modulation depth 0.8. The middle graph shows an m-sequence modulation at depth 1.0. The bottom graph shows a perturbative m-sequence modulation at depth 0.2 added to a sinusoidal modulation at depth 0.8.

## 4   Results

Figure 2 shows the spike rates for both the pure sinusoid and the combined sinusoid and m-sequence stimuli. Note that the rates are nearly the same, indicating that the perturbation did not have a large effect on the average response of the unit. The unit shows an adaptation in firing rate over the 10 trials, but we did not find

a statistically significant change in the kernels of different trials in any of the units.

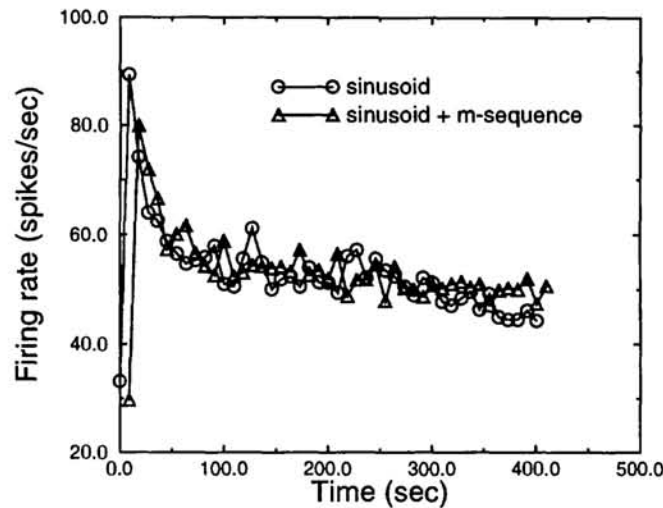

Figure 2: A plot of the unit firing rates for both the pure sinusoid and the sinusoid + m-sequence stimuli. The carrier frequency is 9 kHz and is close to the characteristic frequency of the neuron. The sinusoidal modulation has a frequency of 20 Hz and the m-sequence modulation has a frequency of 800 sec$^{-1}$.

Figure 3 shows modulation response kernels at several different values of the modulation depth. Note that if the system was a linear, superposition would cause all the kernels to be equivalent; in fact it is seen that the nonlinearities are of the same magnitude as the linear response. In this particular unit, the triphasic behavior at small modulation depths gives way to monophasic behavior at high modulation depths and an FFT of the kernel shows that the bandwidth of the modulation transfer function also broadens with increasing depth.

## 5 Discussion

In this paper, we have introduced a new type of stimulus, perturbative m-sequences, for the study of auditory systems and derived their properties. We then applied perturbative m-sequences to the analysis of the modulation response of units in the IC, and found the linear response at a few different operation point. We demonstrated that the nonlinear response in the presence of sinusoidal modulations are nearly as large as the linear response and thus a description of unit response with only an MTF is incomplete. We believe that perturbative stimuli can be an effective tool for the analysis of many systems whose units phase lock to a stimulus.

The main limiting factor is the systematic noise discussed in section 2, but it is possible to trade off duration of measurement and size of the perturbation to achieve good results. The m-sequence stimuli also make it possible to derive higher order information [Sutter1987] and with a suitable noise floor, it may be possible to derive second-order kernels as well.

This work was supported by The Sloan foundation and ONR grant number N00014-94-1-0547.

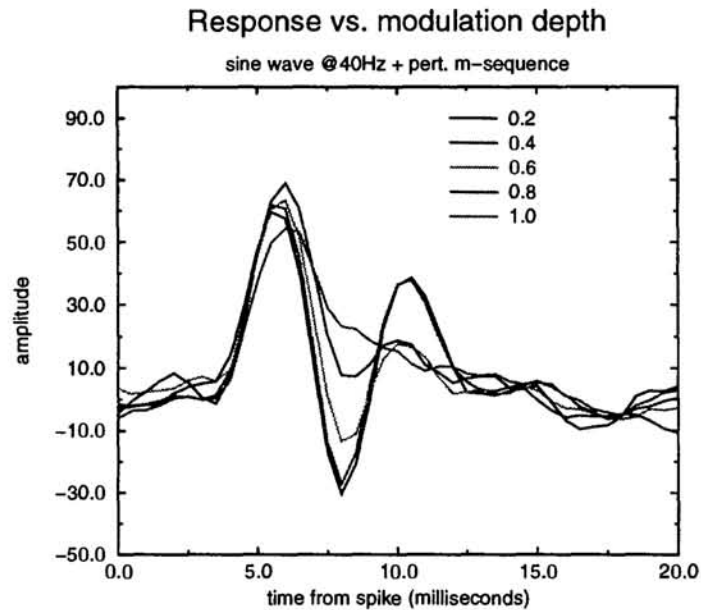

Figure 3: A plot of the temporal kernels derived from perturbative m-sequence stimuli in conjunction with sinusoidal modulations at various modulation depth. The y-axis units are amplitude per spike and the x-axis is in milliseconds *before* the spike.

## Footnotes

*Email: kvale@phy.ucsf.edu and chris@phy.ucsf.edu

# References

[Golomb1982] S. W. Golomb. *Shift Register Sequences*. Aegean Park Press, Laguna Hills, CA, 1982.

[Langner and Schreiner1988] G. Langner and C. E. Schreiner. Periodicity coding in the inferior colliculus of the cat: I. neuronal mechanisms. *Journal of Neurophysiology*, 60:1799–1822, 1988.

[Marmarelis and Marmarelis1978] Panos Z. Marmarelis and Vasilis Z. Marmarelis. *Analysis of Physiological Systems*. Plenum Press, New York, NY, 10011, 1978.

[Møller and Rees1986] Aage R. Møller and Adrian Rees. Dynamic properties of single neurons in the inferior colliculus of the rat. *Hearing Research*, 24:203–215, 1986.

[Pinter and Nabet1987] Robert B. Pinter and Bahram Nabet. *Nonlinear Vision*. CRC Press, Boca Raton, FL, 1987.

[Rees and Møller1987] Adrian Rees and Aage R. Møller. Stimulus properties influencing the responses of inferior colliculus neurons to amplitude-modulated sounds. *Hearing Research*, 27:129–143, 1987.

[Rieke *et al.*1995] F. Rieke, D. A. Bodnar, and W. Bialek. Naturalistic stimuli increase the rate and efficiency of information transmission by primary auditory afferents. *Proceedings of the Royal Society of London. Series B*, 262:259–265, 1995.

[Sutter1987] E. E. Sutter. A practical non-stochastic approach to nonlinear time-domain analysis. In Vasilis Z. Marmarelis, editor, *Advanced Methods of Physiological Modeling, Vol. 1*, pages 303–315. Biomedical Simulations Resource, University of Southern California, Los Angeles, CA 90089-1451, 1987.
